# Generalization and Parameter Estimation in Feedforward Nets: Some Experiments

N. Morgan[†]
[†]International Computer Science Institute
Berkeley, CA 94704, USA

H. Bourlard[†‡]
[‡]Philips Research Laboratory Brussels
B-1170 Brussels, Belgium

## ABSTRACT

We have done an empirical study of the relation of the number of parameters (weights) in a feedforward net to generalization performance. Two experiments are reported. In one, we use simulated data sets with well-controlled parameters, such as the signal-to-noise ratio of continuous-valued data. In the second, we train the network on vector-quantized mel cepstra from real speech samples. In each case, we use back-propagation to train the feedforward net to discriminate in a multiple class pattern classification problem. We report the results of these studies, and show the application of cross-validation techniques to prevent overfitting.

## 1   INTRODUCTION

It is well known that system models which have too many parameters (with respect to the number of measurements) do not generalize well to new measurements. For instance, an autoregressive (AR) model can be derived which will represent the training data with no error by using as many parameters as there are data points. This would

generally be of no value, as it would only represent the training data. Criteria such as the Akaike Information Criterion (AIC) [Akaike, 1974, 1986] can be used to penalize both the complexity of AR models and their training error variance. In feedforward nets, we do not currently have such a measure. In fact, given the aim of building systems which are biologically plausible, there is a temptation to assume the usefulness of indefinitely large adaptive networks. In contrast to our best guess at Nature's tricks, man-made systems for pattern recognition seem to require nasty amounts of data for training. In short, the design of massively parallel systems is limited by the number of parameters that can be learned with available training data. It is likely that the only way truly massive systems can be built is with the help of prior information, e.g., connection topology and weights that need not be learned [Feldman et al, 1988].

Learning theory [Valiant, V.N., 1984; Pearl, J., 1978] has begun to establish what is possible for trained systems. Order-of-magnitude lower bounds have been established for the number of required measurements to train a desired size feedforward net [Baum&Haussler, 1988]. Rules of thumb suggesting the number of samples required for specific distributions could be useful for practical problems. Widrow has suggested having a training sample size that is 10 times the number of weights in a network ("Uncle Bernie's Rule")[Widrow, 1987]. We have begun an empirical study of the relation of the number of parameters in a feedforward net (e.g. hidden units, connections, feature dimension) to generalization performance for data sets with known discrimination complexity and signal-to-noise ratio. In the experiment reported here, we are using simulated data sets with controlled parameters, such as the number of clusters of continuous-valued data. In a related practical example, we have trained a feedforward network on vector-quantized mel cepstra from real speech samples. In each case, we are using the back-propagation algorithm [Rumelhart et al, 1986] to train the feedforward net to discriminate in a multiple class pattern classification problem. Our results confirm that estimating more parameters than there are training samples can degrade generalization. However, the peak in generalization performance (for the difficult pattern recognition problems tested here) can be quite broad if the networks are not trained too long, suggesting that previous guidelines for network size may have been conservative. Furthermore, cross-validation techniques, which have also proved quite useful for autoregressive model order determination, appear to improve generalization when used as a stopping criterion for iteration, and thus preventing overtraining.

## 2  RANDOM VECTOR PROBLEM

### 2.1  METHODS

Studies based on synthesized data sets will generally show behavior that is different from that seen with a real data set. Nonetheless, such studies are useful because of the ease with which variables of interest may be altered. In this case, the object was to manufacture a difficult pattern recognition problem with statistically regular variability between the training and test sets. This is actually no easy trick; if the problem is too easy, then even very small nets will be sufficient, and we would not be modeling the

problem of doing hard pattern classification with small amounts of training data. If the problem is too hard, then variations in performance will be lost in the statistical variations inherent to methods like back-propagation, which use random initial weight values.

Random points in a 4-dimensional hyperrectangle (drawn from a uniform probability distribution) are classified arbitrarily into one of 16 classes. This group of points will be referred to as a cluster. This process is repeated for 1-4 nonoverlapping hyperrectangles. A total of 64 points are chosen, 4 for each class. All points are then randomly perturbed with noise of uniform density and range specified by a desired signal-to-noise ratio (SNR). The noise is added twice to create 2 data sets, one to be used for training, and the other for test. Intuitively, one might expect that 16-64 hidden units would be required to transform the training space for classification by the output layer. However, the variation between training and test and the relatively small amount of data (256 numbers) suggest that for large numbers of parameters (over 256) there should be a significant degrading of generalization. Another issue was how performance in such a situation would vary over large numbers of iterations.

Simulations were run on this data using multi-layer perceptrons(MLP) (i.e., layered feedforward networks) with 4 continuous-valued inputs, 16 outputs, and a hidden layer of sizes ranging from 4 to 128. Nets were run for signal-to-noise ratios of 1.0 and 2.0, where the SNR is defined as the ratio of the range of the original cluster points to the range of the added random values. Error back-propagation without momentum was used, with an adaptation constant of .25 . For each case, the 64 training patterns were used 10,000 times, and the resulting network was tested on the second data set every 100 iterations so that generalization could be observed during the learning. Blocks of ten scores were averaged to stabilize the generalization estimate. After this smoothing, the standard deviation of error (using the normal approximation to the binomial distribution) was roughly 1%. Therefore, differences of 3% in generalization performance are significant at a level of .001 . All computation was performed on Sun4-110's using code written in C at ICSI. Roughly a trillion floating point operations were required for the study.

## 2.2  RESULTS

Table I shows the test performance for a single cluster and a signal-to-noise ratio of 1.0 . The chart shows the variation over a range of iterations and network size (specified both as #hidden units, and as ratio of #weights to #measurements, or "weight ratio"). Note that the percentages can have finer gradation than 1/64, due to the averaging, and that the performance on the training set is given in parentheses. Test performance is best for this case for 8 hidden units (24.7%), or a weight ratio of .62 (after 2000 iterations), and for 16 units (21.9%), or a weight ratio of 1.25 (after 10000 iterations). For larger networks, the performance degrades, presumably because of the added noise. At 2000 iterations, the degradation is statistically significant, even in going from 8 to 16 hidden units. There is further degradation out to the 128-unit case. The surprising thing is that, while this degradation is quite noticeable, it is quite graceful considering the order-of magnitude range in net sizes. An even stronger effect is the loss of generalization power when the larger nets are more fully trained. All of the nets generalized better when

they were trained to a relatively poor degree, especially the larger ones.

Table I — Test (and training) scores: 1 cluster, SNR = 1.0

| #hidden units | #weights / #inputs | %Test (Train) Correct after N Iterations | | | |
|---|---|---|---|---|---|
| | | 1000 | 2000 | 5000 | 10000 |
| 4 | .31 | 9.2(4.4) | 21.7(15.6) | 12.0(25.9) | 15.6(34.4) |
| 8 | .62 | 11.4(5.2) | 24.7(17.0) | 20.6(29.8) | 21.4(63.9) |
| 16 | 1.25 | 13.6(6.9) | 21.1(18.4) | 18.3(37.2) | 21.9(73.4) |
| 32 | 2.50 | 12.8(6.4) | 18.4(18.3) | 17.8(41.7) | 13.0(80.8) |
| 64 | 5.0 | 13.6(7.7) | 18.3(20.8) | 19.7(34.4) | 18.0(79.2) |
| 128 | 10.0 | 11.6(6.7) | 17.7(19.1) | 12.2(34.7) | 15.6(75.6) |

Table II shows the results for the same 1-cluster problem, but with higher SNR data (2.0). In this case, a higher level of test performance was reached, and it was reached for a larger net with more iterations (40.8% for 64 hidden units after 5000 iterations). At this point in the iterations, no real degradation was seen for up to 10 times the number of weights as data samples. However, some signs of performance loss for the largest nets was evident after 10000 iterations. Note that after 5000 iterations, the networks were only half-trained (roughly 50% error on the training set). When they were 80-90% trained, the larger nets lost considerable ground. For instance, the 10 x net (128 hidden units) lost performance from 40.5% to 28.1% during these iterations. It appears that the higher signal-to-noise of this example permitted performance gains for even higher overparametrization factors, but that the result was even more sensitive to training for too many iterations.

Table II — Test (and training) scores: 1 cluster, SNR = 2.0

| #hidden units | #weights / #inputs | %Test (Train) Correct after N Iterations | | | |
|---|---|---|---|---|---|
| | | 1000 | 2000 | 5000 | 10000 |
| 4 | .31 | 18.1(8.4) | 25.6(29.1) | 32.2(29.8) | 26.9(29.2) |
| 8 | .62 | 22.5(12.8) | 31.1(34.7) | 34.5(44.5) | 33.3(62.2) |
| 16 | 1.25 | 22.0(11.6) | 33.4(32.8) | 33.6(57.2) | 29.4(78.3) |
| 32 | 2.50 | 25.6(13.3) | 33.4(35.2) | 39.4(51.1) | 34.2(87.0) |
| 64 | 5.0 | 26.4(13.9) | 36.1(35.0) | 40.8(45.2) | 33.6(86.9) |
| 128 | 10.0 | 26.9(12.0) | 34.5(34.5) | 40.5(47.2) | 28.1(91.1) |

Table III shows the performance for a 4-cluster case, with SNR = 1.0 . Small nets are omitted here, because earlier experiments showed this problem to be too hard. The best performance (21.1%) is for one of the larger nets at 2000 iterations, so that the degradation effect is not clearly visible for the undertrained case. At 10000 iterations, however, the larger nets do poorly.

Table III — Test (and training) scores: 4 cluster, SNR = 1.0

| #hidden units | #weights #inputs | %Test (Train) Correct after N Iterations | | | |
|---|---|---|---|---|---|
| | | 1000 | 2000 | 5000 | 10000 |
| 32 | 2.50 | 13.8(12.7) | 18.3(23.6) | 15.8(38.8) | 9.4(71.4) |
| 64 | 5.0 | 13.6(12.7) | 18.4(23.6) | 14.7(42.7) | 18.8(71.6) |
| 96 | 7.5 | 15.3(13.0) | 21.1(24.7) | 15.9(45.5) | 16.3(78.1) |
| 128 | 10. | 15.2(13.1) | 19.1(23.8) | 17.5(40.5) | 10.5(70.9) |

Figure 1 illustrates this graphically. The "undertrained" case is relatively insensitive to the network size, as well as having the highest raw score.

## 3   SPEECH RECOGNITION

### 3.1   METHODS

In an ongoing project at ICSI and Philips, a German language data base consisting of 100 training and 100 test sentences (both from the same speaker) were used for training of a multi-layer-perceptron (MLP) for recognition of phones at the frame level, as well as to estimate probabilities for use in the dynamic programming algorithm for a discrete Hidden Markov Model (HMM) [Bourlard & Wellekens, 1988; Bourlard et al, 1989]. Vector-quantized mel cepstra were used as binary input to a hidden layer. Multiple frames were used as input to provide context to the network. While the size of the output layer was kept fixed at 50 units, corresponding to the 50 phonemes to be recognized, the hidden layer was varied from 20 to 200 units, and the input context was kept fixed at 9 frames of speech. As the acoustic vectors were coded on the basis of 132 prototype vectors by a simple binary vector with only one bit 'on', the input field contained 9x132=1188 units, and the total number of possible inputs was thus equal to $132^9$. There were 26767 training patterns and 26702 independent test patterns. Of course, this represented only a very small fraction of the possible inputs, and generalization was thus potentially difficult. Training was done by the classical "error-back propagation" algorithm, starting by minimizing an entropy criterion [Solla et al, 1988] and then the standard least-mean-square error (LMSE) criterion. In each iteration, the complete training set was presented, and the parameters were updated after each training pattern.

To avoid overtraining of the MLP, (as was later demonstrated by the random vector experiment described above), improvement on the test set was checked after each iteration. If the classification rate on the test set was decreasing, the adaptation parameter of the gradient procedure was decreased, otherwise it was kept constant. In another experiment this approach was systematized by splitting the data in three parts: one for the training, one for the test and a third one absolutely independent of the training procedure for validation. No significant difference was observed between classification rates for the test and validation data.

Other than the obvious difference with the previous study (this used real data), it is important to note another significant point: in this case, we stopped iterating (by any one particular criterion) when that criterion was leading to no new test set performance improvement. While we had not yet done the simulations described above, we had observed the necessity for such an approach over the course of our speech research. We expected this to ameliorate the effects of overparameterization.

## 3.2    RESULTS

Table IV shows the variation in performance for 5, 20, 50, and 200 hidden units. The peak at 20 hidden units for test set performance, in contrast to the continued improvement in training set performance, can be clearly seen. However, the effect is certainly a mild one given the wide range in network size; using 10 times the number of weights as in the "peak" case only causes a degradation of 3.1%. Note, however, that for this experiment, the more sophisticated training procedure was used which halted training when generalization started to degrade.

For comparison with classical approaches, results obtained with Maximum Likelihood (ML) and Bayes estimates are also given. In those cases, it is not possible to use contextual information, because the number of parameters to be learned would be $50 * 132^9$ for the 9 frames of context. Therefore, the input field was restricted to a single frame. The number of parameters for these two last classifiers was then $50 * 132 = 6600$, or a parameter/measurement ratio of .25 . This restriction explains why the Bayes classifier, which is inherently optimal for a given pattern classification problem, is shown here as yielding a lower performance than the potentially suboptimal MLP.

Table IV — Test Run: Phoneme Recognition on German data base

| hidden units | #parameters/#training_numbers | training | test |
|:---:|:---:|:---:|:---:|
| 5 | .23 | 62.8 | 54.2 |
| 20 | .93 | 75.7 | 62.7 |
| 50 | 2.31 | 73.7 | 60.6 |
| 200 | 9.3 | 86.7 | 59.6 |
| ML | .25 | 45.9 | 44.8 |
| Bayes | .25 | 53.8 | 53.0 |

# 4  CONCLUSIONS

While both studies show the expected effects of overparameterization, (poor generalization, sensitivity to overtraining in the presence of noise), perhaps the most significant result is that it was possible to greatly reduce the sensitivity to the choice of network size by directly observing the network performance on an independent test set during the course of learning (cross-validation). If iterations are not continued past this point, fewer measurements are required. This only makes sense because of the interdependence of the learned parameters, particularly for the undertrained case. In any event, though, it is clear that adding parameters over the number required for discrimination is wasteful of resources. Networks which require many more parameters than there are measurements will certainly reach lower levels of peak performance than simpler systems. For at least the examples described here, it is clear that both the size of the MLP and the degree to which it should be trained are parameters which must be learned from experimentation with the data set. Further study might, perhaps, yield enough results to permit some rule of thumb dependent on properties of the data, but our current thinking is that these parameters should be determined dynamically by testing on an independent test set.

## References

Akaike, H. (1974), "A new look at the statistical model identification." *IEEE Trans. autom. Control*, AC-10, 667-674

Akaike, H. (1986), "Use of Statistical Models for Time Series Analysis", Vol. 4, Proc. IEEE Intl. Conference on Acoustics, Speech, and Signal Processing, Tokyo, 1986, pp.3147-3155

Baum, E.B., & Haussler, D., (1988), "What Size Net Gives Valid Generalization?", Neural Computation, In Press

Bourlard, H., Morgan, N., & Wellekens, C.J., (1989), "Statistical Inference in Multilayer Perceptrons and Hidden Markov Models, with Applications in Continuous Speech Recognition", NATO Advanced Research Workshop, Les Arcs, France

Feldman, J.A., Fanty, M.A., and Goddard, N., (1988) "Computing with Structured Neural Networks", Computer, vol. 21, No.3, pp 91-104

Pearl,J., (1978), "On the Connection Between the Complexity and Credibility of Inferred Models", Int. J. General Systems, Vol.4, pp. 155-164

Rumelhart, D.E., Hinton, G.E., & Williams, R.J., (1986), "Learning internal representations by error propagation" in *Parallel Distributed Processing* (D.E. Rumelhart & J.L. McClelland, Eds.), ch. 15, Cambridge, MA: MIT Press

Valiant, L.G., (1984), "A theory of the learnable", Comm. ACM V27, N11 pp1134-1142

Widrow, B, (1987) "ADALINE and MADALINE" , Plenary Speech, Vol. I, Proc. IEEE 1st Intl. Conf. on Neural Networks, San Diego, CA, 143-158

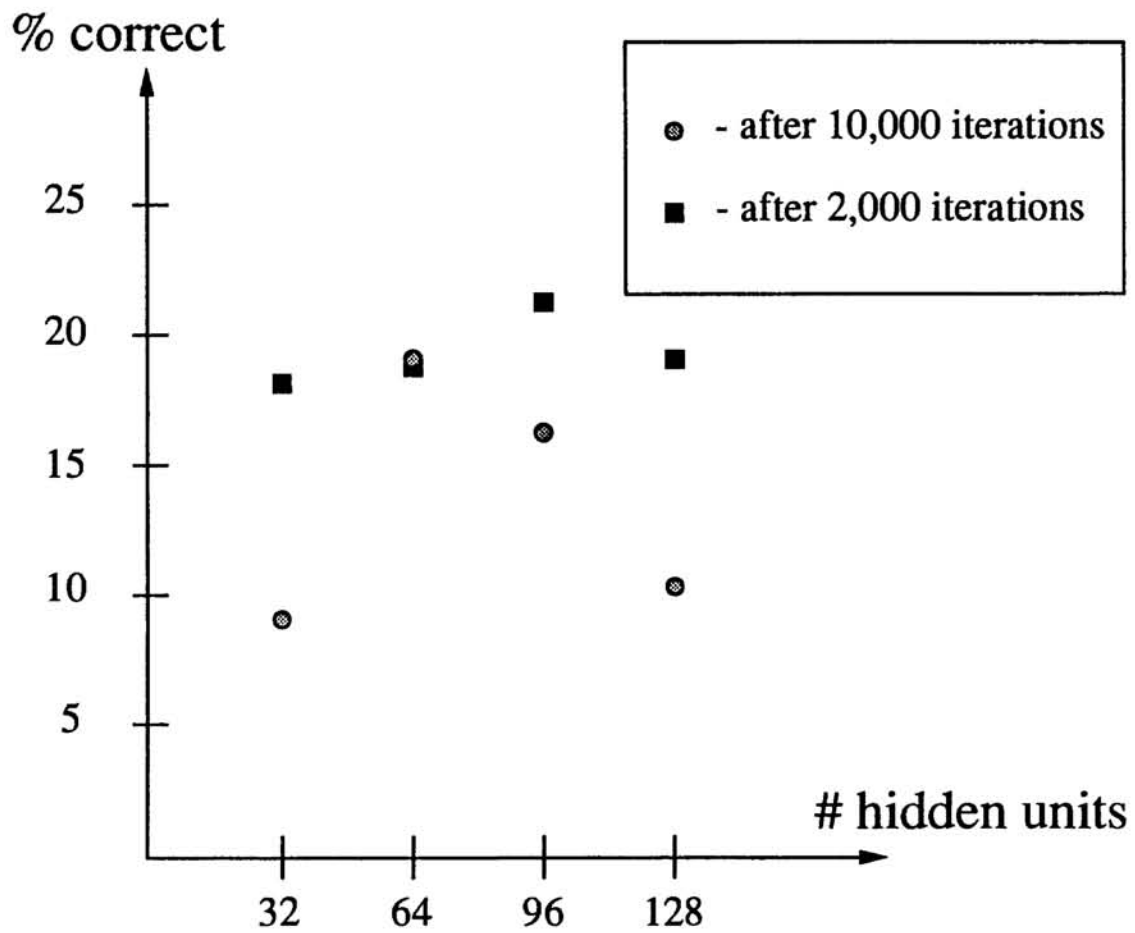

Figure 1: Sensitivity to net size